# Incrementally Learning Time-varying Half-planes

**Anthony Kuh***
Dept. of Electrical Engineering
University of Hawaii at Manoa
Honolulu, HI 96822

**Thomas Petsche**[†]
Siemens Corporate Research
755 College Road East
Princeton, NJ 08540

**Ronald L. Rivest**[‡]
Laboratory for Computer Science
MIT
Cambridge, MA 02139

## Abstract

We present a distribution-free model for incremental learning when concepts vary with time. Concepts are caused to change by an adversary while an incremental learning algorithm attempts to track the changing concepts by minimizing the error between the current target concept and the hypothesis. For a single half-plane and the intersection of two half-planes, we show that the average mistake rate depends on the maximum rate at which an adversary can modify the concept. These theoretical predictions are verified with simulations of several learning algorithms including back propagation.

## 1 INTRODUCTION

The goal of our research is to better understand the problem of learning when concepts are allowed to change over time. For a dichotomy, concept drift means that the classification function changes over time. We want to extend the theoretical analyses of learning to include time-varying concepts; to explore the behavior of current learning algorithms in the face of concept drift; and to devise tracking algorithms to better handle concept drift. In this paper, we briefly describe our theoretical model and then present the results of simulations

*kuh@wiliki.eng.hawaii.edu    †petsche@learning.siemens.com    ‡rivest@theory.lcs.mit.edu

in which several tracking algorithms, including an on-line version of back-propagation, are applied to time-varying half-spaces.

For many interesting real world applications, the concept to be learned or estimated is not static, i.e., it can change over time. For example, a speaker's voice may change due to fatigue, illness, stress or background noise (Galletti and Abbott, 1989), as can handwriting. The output of a sensor may drift as the components age or as the temperature changes. In control applications, the behavior of a plant may change over time and require incremental modifications to the model.

Haussler, *et al.* (1987) and Littlestone (1989) have derived bounds on the number of mistakes an on-line learning algorithm will make while learning any concept in a given concept class. However, in that and most other learning theory research, the concept is assumed to be fixed. Helmbold and Long (1991) consider the problem of concept drift, but their results apply to memory-based tracking algorithms while ours apply to incremental algorithms. In addition, we consider different types of adversaries and use different methods of analysis.

## 2   DEFINITIONS

We use much the same notation as most learning theory, but we augment many symbols with a subscript to denote time. As usual, $X$ is the instance space and $x_t$ is an instance drawn at time $t$ according to a *fixed*, arbitrary distribution $P_X$. The function $c_t : X \rightarrow \{0,1\}$ is the active concept at time $t$, that is, at time $t$ any instance is labeled according to $c_t$. The label of the instance is $a_t = c_t(x_t)$. Each active concept $c_i$ is a member of the concept class $C$. A sequence of active concepts is denoted $\mathbf{c}$. At any time $t$, the tracker uses an algorithm $\mathcal{L}$ to generate a hypothesis $\hat{c}_t$ of the active concept.

We use a symmetric distance function to measure the difference between two concepts: $d(c, c') = P_X[x : c(x) \neq c'(x)]$.

As we alluded to in the introduction, we distinguish between two types of tracking algorithms. A *memory-based* tracker stores the most recent $m$ examples and chooses a hypothesis based on those stored examples. Helmbold and Long (1991), for example, use an algorithm that chooses as the hypothesis the concept that minimizes the number of disagreements between $\hat{c}_t(x_t)$ and $c_t(x_t)$. An *incremental* tracker uses only the previous hypothesis and the most recent examples to form the new hypothesis. In what follows, we focus on incremental trackers.

The task for a tracking algorithm is, at each iteration t, to form a "good" estimate $\hat{c}_t$ of the active concept $c_t$ using the sequence of previous examples. Here "good" means that the probability of a disagreement between the label predicted by the tracker and the actual label is small. In the time-invariant case, this would mean that the tracker would incrementally improve its hypothesis as it collects more examples. In the time-varying case, however, we introduce an adversary whose task is to change the active concept at each iteration.

Given the existence of a tracker and an adversary, each iteration of the tracking problem consists of five steps: (1) the adversary chooses the active concept $c_t$; (2) the tracker is given an unlabeled instance, $x_t$, chosen randomly according to $P_X$; (3) the tracker predicts a label using the current hypothesis: $\hat{a}_t = \hat{c}_{t-1}(x_t)$; (4) the tracker is given the correct label $a_t = c_t(x_t)$; (5) the tracker forms a new hypothesis: $\hat{c}_t = \mathcal{L}(\hat{c}_{t-1}, \langle x_t, a_t \rangle)$.

It is clear that an unrestricted adversary can always choose a concept sequence (a sequence of active concepts) that the tracker can not track. Therefore, it is necessary to restrict the changes that the adversary can induce. In this paper, we require that two subsequent concepts differ by no more than $\gamma$, that is, $d(c_t, c_{t-1}) \leq \gamma$ for all $t$. We define the restricted concept sequence space $C_\gamma = \{c : c_t \in C, d(c_t, c_{t+1}) \leq \gamma\}$. In the following, we are concerned with two types of adversaries: a *benign* adversary which causes changes that are independent of the hypothesis; and a *greedy* adversary which always chooses a change that will maximize $d(c_t, c_{t-1})$ constrained by the upper-bound.

Since we have restricted the adversary, it seems only fair to restrict the tracker too. We require that a tracking algorithm be: *deterministic*, i.e., that the process generating the hypotheses be deterministic; *prudent*, i.e., that the label predicted for an instance be a deterministic function of the current hypothesis: $\hat{a}_t = \hat{c}_{t-1}(x_t)$; and *conservative*, i.e., that the hypothesis is modified only when an example is mislabeled. The restriction that a tracker be conservative rules out algorithms which attempt to predict the adversary's movements and is the most restrictive of the three. On the other hand, when the tracker does update its hypothesis, there are no restrictions on $d(\hat{c}_t, \hat{c}_{t-1})$.

To measure performance, we focus on the mistake rate of the tracker. A *mistake* occurs when the tracker mislabels an instance, i.e., whenever $\hat{c}_{t-1}(x_t) \neq c_t(x_t)$. For convenience, we define a mistake indicator function, $M(x_t, c_t, \hat{c}_{t-1})$ which is 1 if $\hat{c}_{t-1}(x_t) \neq c_t(x_t)$ and 0 otherwise. Note that if a mistake occurs, it occurs before the hypothesis is updated — a conservative tracker is always a step behind the adversary. We are interested in the *asymptotic mistake rate*, $\mu = \liminf_{t \to \infty} \frac{1}{t} \sum_{i=0}^{t} M(x_t, c_t, \hat{c}_{t-1})$.

Following Helmbold and Long (1991), we say that an algorithm $(\mu, \gamma)$-tracks a sequence space $C$ if, for all $c \in C_\gamma$ and all drift rates $\gamma'$ not greater than $\gamma$, the mistake rate $\mu'$ is at most $\mu$.

We are interested in bounding the asymptotic mistake rate of a tracking algorithm based on the concept class and the adversary. To derive a lower bound on the mistake rate, we hypothesize the existence of a perfect conservative tracker, i.e., one that is always able to guess the correct concept each time it makes a mistake. We say that such a tracker has *complete side information* (CSI). No conservative tracker can do better than one with CSI. Thus, the mistake rate for a tracker with CSI is a lower bound on the mistake rate achievable by any conservative tracker.

To upper bound the mistake rate, it is necessary that we hypothesize a particular tracking algorithm when *no side information* (NSI) is available, that is, when the tracker only knows it mislabeled an instance and nothing else. In our analysis, we study a simple tracking algorithm which modifies the previous hypothesis just enough to correct the mistake.

## 3   ANALYSIS

We consider two concept classes in this paper, half-planes and the intersection of two half-planes which can be defined by lines in the plane that pass through the origin. We call these classes $\mathsf{HS}_2$ and $\mathsf{IHS}_2$. In this section, we present our analysis for $\mathsf{HS}_2$.

Without loss of generality, since the lines pass through the origin, we take the instance space to be the circumference of the unit circle. A half-plane in $\mathsf{HS}_2$ is defined by a vector $\mathbf{w}$ such that for an instance $x$, $c(x) = 1$ if $\mathbf{w}x \geq 0$ and $c(x) = 0$ otherwise. Without loss of

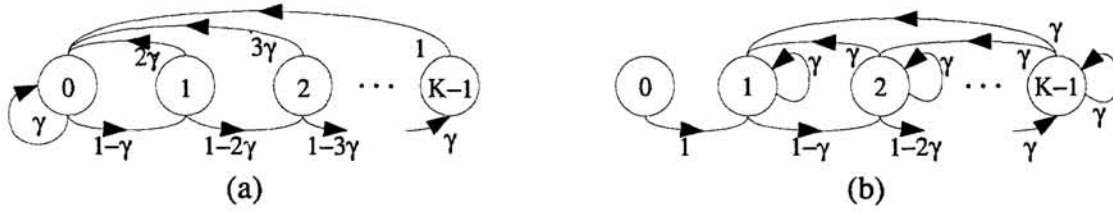

Figure 1: Markov chain for the greedy adversary and (a) CSI and (b) COVER trackers.

generality, as we will show later, we assume that the instances are chosen uniformly.

To begin, we assume a greedy adversary as follows: Every time the tracker guesses the correct target concept (that is, $\hat{c}_{t-1} = c_{t-1}$), the greedy adversary randomly chooses a vector $r$ orthogonal to $w$ and at every iteration, the adversary rotates $w$ by $\pi\gamma$ radians in the direction defined by $r$. We have shown that a greedy adversary maximizes the asymptotic mistake rate for a conservative tracker but do not present the proof here.

To lower bound the achievable error rate, we assume a conservative tracker with complete side information so that the hypothesis is unchanged if no mistake occurs and is updated to the correct concept otherwise. The state of this system is fully described by $d(c_t, \hat{c}_t)$ and, for $\gamma = 1/K$ for some integer $K$, is modeled by the Markov chain shown in figure 1a. In each state $s_i$ (labeled $i$ in the figure), $d(c_t, \hat{c}_t) = i\gamma$. The asymptotic mistake rate is equal to the probability of state 0 which is lower bounded by

$$l(\gamma) = \sqrt{2\gamma/\pi} - 2\gamma/\pi$$

Since $l(\gamma)$ depends only on $\gamma$ which, in turn, is defined in terms of the probability measure, the results holds for all distributions. Therefore, since this result applies to the best of all possible conservative trackers, we can say that

**Theorem 1.** *For* $HS_2$*, if* $d(c_t, c_{t-1}) \leq \gamma$*, then there exists a concept sequence* $c \in C_\gamma$ *such that the mistake rate* $\mu > l(\gamma)$*. Equivalently,* $C_\gamma$ *is not* $(\gamma, \mu)$*-trackable whenever* $\mu < l(\gamma)$*.*

To upper bound the achievable mistake rate, we must choose a realizable tracking algorithm. We have analyzed the behavior of a simple algorithm we call COVER which rotates the hypothesize line just far enough to cover the incorrectly labeled instance. Mathematically, if $\hat{w}_t$ is the hypothesized normal vector at time $t$ and $x_t$ is the mislabeled instance:

$$\hat{w}_t = \hat{w}_{t-1} - (x_t \cdot \hat{w}_{t-1})x_t. \tag{1}$$

In this case, a mistake in state $s_i$ can lead to a transition to any state $s_j$ for $j \leq i$ as shown in Figure 1b. The asymptotic probability of a mistake is the sum of the equilibrium transition probabilities $P(s_j|s_i)$ for all $j \leq i$. Solving for these probabilities leads to an upper bound $u(\gamma)$ on the mistake rate:

$$u(\gamma) = \sqrt{\pi\gamma/2} + \gamma(2 + \sqrt{1/e})$$

Again this depends only on $\gamma$ and so is distribution independent and we can say that:

**Theorem 2.** *For* $HS_2$*, for all concept sequences* $c \in C_\gamma$ *the mistake rate for* COVER $\mu \leq u(\gamma)$*. Equivalently,* $C_\gamma$ *is* $(\gamma, \mu)$*-trackable whenever* $\mu < u(\gamma)$*.*

If the adversary is benign, it is as likely to decrease as to increase the probability of a mistake. Unfortunately, although this makes the task of the tracker easier, it also makes the analysis more difficult. So far, we can show that:

**Theorem 3.** *For* $\text{HS}_2$ *and a benign adversary, there exists a concept sequence* $c \in C_\gamma$ *such that the mistake rate* $\mu$ *is* $O(\gamma^{2/3})$.

## 4  SIMULATIONS

To test the predictions of the theory and explore some areas for which we currently have no theory, we have run simulations for a variety of concept classes, adversaries, and tracking algorithms. Here we will present the results for single half-planes and the intersection of two half-planes; both greedy and benign adversaries; an ideal tracker; and two types of trackers that use no side information.

### 4.1  HALF-PLANES

The simplest concept class we have simulated is the set of all half-planes defined by lines passing through the origin. This is equivalent to the set classifications realizable with 2-dimensional perceptrons with zero threshold. In other words, if $w$ is the normal vector and $x$ is a point in space, $c(x) = 1$ if $w \cdot x \geq 0$ and $c(x) = 0$ otherwise. The mistake rate reported for each data point is the average of 1,000,000 iterations. The instances were chosen uniformly from the circumference of the unit circle.

We also simulated the ideal tracker using an algorithm called CSI and tested a tracking algorithm called COVER, which is a simple implementation of the tracking algorithm analyzed in the theory. If a tracker using COVER mislabels an instance, it rotates the normal vector in the plane defined by it and the instance so that the instance lies exactly on the new hypothesis line, as described by equation 1.

#### 4.1.1  Greedy adversary

Whenever CSI or COVER makes a mistake and then guesses the concept exactly, the greedy adversary uniformly at random chooses a direction orthogonal to the normal vector of the hyperplane. Whenever COVER makes a mistake and $\widehat{w}_t \neq w_t$, the greedy adversary choose the rotation direction to be in the plane defined by $w_t$ and $\widehat{w}_t$ and orthogonal to $w_t$. At every iteration, the adversary rotates the normal vector of the hyperplane in the most recently chosen direction so that $d(c_t, c_{t+1}) = \gamma$, or equivalently, $w_t \cdot w_{t-1} = \cos(\pi\gamma)$.

Figure 2 shows that the theoretical lower bound very closely matches the simulation results for CSI when $\gamma$ is small. For small $\gamma$, the simulation results for COVER lie very close to the theoretical predictions for the NSI case. In other words, the bounds predicted in theorems 1 and 2 are tight and the mistake rates for CSI and COVER differ by only a factor of $\pi/2$.

#### 4.1.2  Benign adversary

At every iteration, the benign adversary uniformly at random chooses a direction orthogonal to the normal vector of the hyperplane and rotates the hyperplane in that direction so that $d(c_t, c_{t+1}) = \gamma$. Figure 3 shows that CSI behaves as predicted by Theorem 3 when $\mu = 0.6\gamma^{2/3}$. The figure also shows that COVER performs very well compared to CSI.

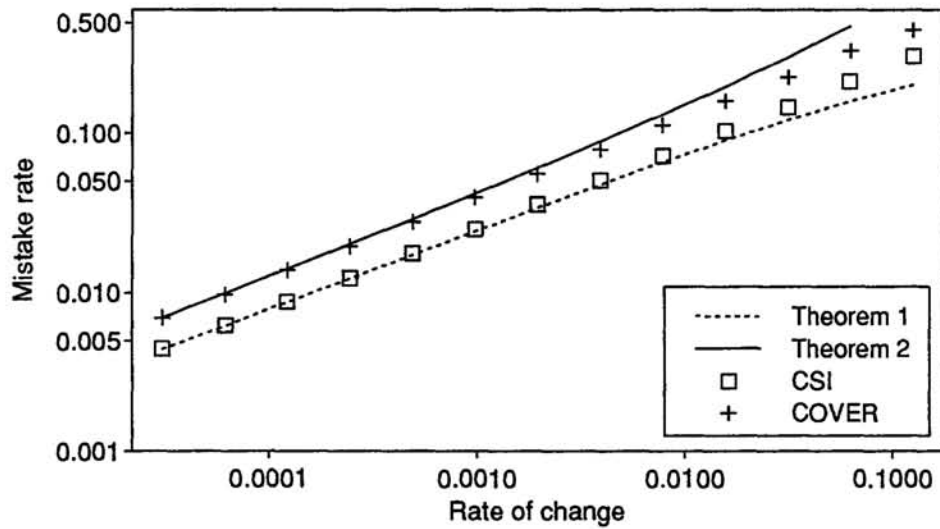

Figure 2: The mistake rate, $\mu$, as a function of the rate of change, $\gamma$, for $HS_2$ when the adversary is greedy.

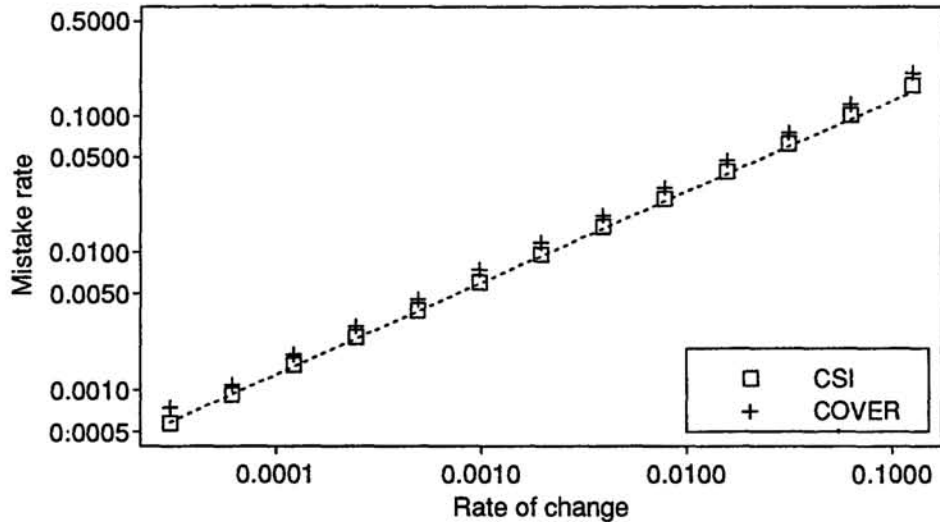

Figure 3: The mistake rate, $\mu$, as a function of the rate of change, $\gamma$, for $HS_2$ when the adversary is benign. The line is $\mu = 0.6\gamma^{2/3}$.

## 4.2    INTERSECTION OF TWO HALF-PLANES

The other concept class we consider here is the intersection of two half-spaces defined by lines through the origin. That is, $c(x) = 1$ if $w_1 x \geq 0$ and $w_2 x \geq 0$ and $c(x) = 0$ otherwise. We tested two tracking algorithms using no side information for this concept class.

The first is a variation on the previous COVER algorithm. For each mislabeled instance: if both half-spaces label $x_t$ differently than $c_t(x_t)$, then the line that is closest in euclidean distance to $x_t$ is updated according to COVER; otherwise, the half-space labeling $x_t$ differently than $c_t(x_t)$ is updated.

The second is a feed-forward network with 2 input, 2 hidden and 1 output nodes. The

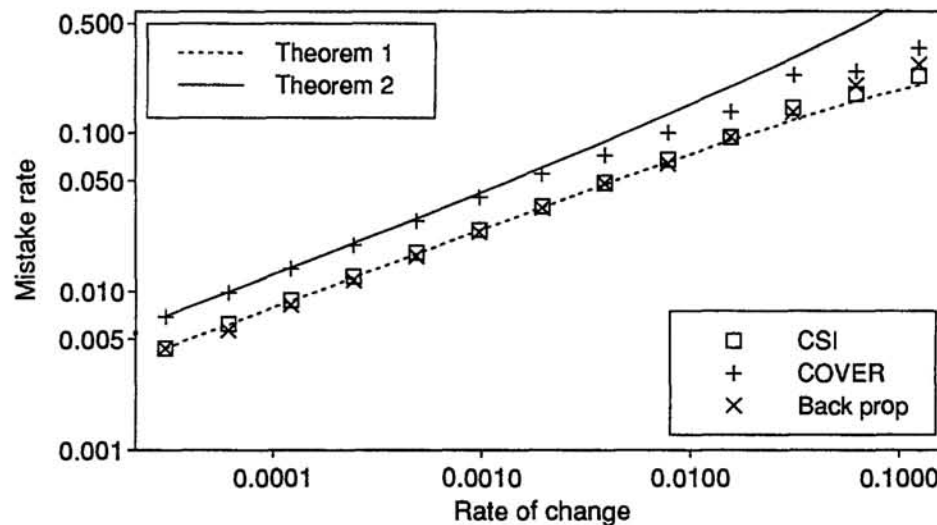

Figure 4: The mistake rate, $\mu$, as a function of the rate of change, $\gamma$, for IHS$_2$ when the adversary is greedy.

thresholds of all the neurons and the weights from the hidden to output layers are fixed, i.e., only the input weights can be modified. The output of each neuron is $f(\mathbf{u}) = (1+e^{-10\mathbf{w}\mathbf{u}})^{-1}$. For classification, the instance was labeled one if the output of the network was greater than 0.5 and zero otherwise. If the difference between the actual and desired outputs was greater than 0.1, back-propagation was run using only the most recent example until the difference was below 0.1. The learning rate was fixed at 0.01 and no momentum was used. Since the model may be updated without making a mistake, this algorithm is not conservative.

### 4.2.1  Greedy Adversary

At each iteration, the greedy adversary rotates each hyperplane in a direction orthogonal to its normal vector. Each rotation direction is based on an initial direction chosen uniformly at random from the set of vectors orthogonal to the normal vector. At each iteration, both the normal vector and the rotation vector are rotated $\pi\gamma/2$ radians in the plane they define so that $d(c_t, c_{t-1}) = \gamma$ for every iteration. Figure 4 shows that the simulations match the predictions well for small $\gamma$. Non-conservative back-propagation performs about as well as conservative CSI and slightly better than conservative COVER.

### 4.2.2  Benign Adversary

At each iteration, the benign adversary uniformly at random chooses a direction orthogonal to $\mathbf{w}_i$ and rotates the hyperplane in that direction such that $d(c_t, c_{t-1}) = \gamma$. The theory for the benign adversary in this case is not yet fully developed, but figure 5 shows that the simulations approximate the optimal performance for HS$_2$ against a benign adversary with $c \in \mathcal{C}_{\gamma/2}$. Non-conservative back-propagation does not perform as well for very small $\gamma$, but catches up for $\gamma > .001$. This is likely due to the particular choice of learning rate.

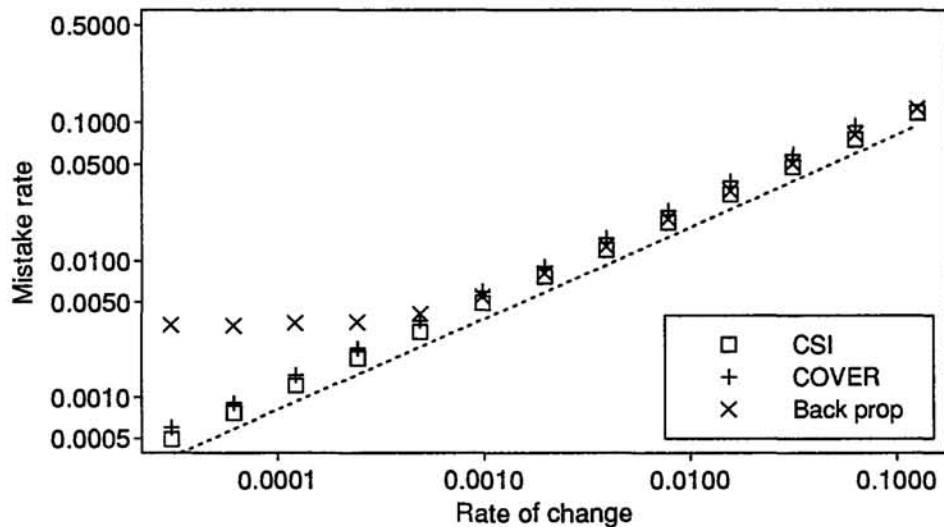

Figure 5: The mistake rate, $\mu$, as a function of the rate of change, $\gamma$, for $\text{IHS}_2$ when the adversary is benign. The dashed line is $\mu = 0.6(\gamma/2)^{2/3}$.

## 5   CONCLUSIONS

We have presented the results of some of our research applied to the problem of tracking time-varying half-spaces. For $\text{HS}_2$ and $\text{IHS}_2$ presented here, simulation results match the theory quite well. For $\text{IHS}_2$, non-conservative back-propagation performs quite well.

We have extended the theorems presented in this paper to higher-dimensional input vectors and more general geometric concept classes. In Theorem 3, $\mu \geq c\gamma^{2/3}$ for some constant $c$ and we are working to find a good value for that constant. We are also working to develop an analysis of non-conservative trackers and to better understand the difference between conservative and non-conservative algorithms.

### Acknowledgments

Anthony Kuh gratefully acknowledges the support of the National Science Foundation through grant EET-8857711 and Siemens Corporate Research. Ronald L. Rivest gratefully acknowledges support from NSF grant CCR-8914428, ARO grant N00014-89-J-1988 and a grant from the Siemens Corporation.

### References

Galletti, I. and Abbott, M. (1989). Development of an advanced airborne speech recognizer for direct voice input. *Speech Technology*, pages 60–63.

Haussler, D., Littlestone, N., and Warmuth, M. K. (1987). Expected mistake bounds for on-line learning algorithms. (Unpublished).

Helmbold, D. P. and Long, P. M. (1991). Tracking drifting concepts using random examples. In Valiant, L. G. and Warmuth, M. K., editors, *Proceedings of the Fourth Annual Workshop on Computational Learning Theory*, pages 13–23. Morgan Kaufmann.

Littlestone, N. (1989). Mistake bounds and logarithmic linear-threshold learning algorithms. Technical Report UCSC-CRL-89-11, Univ. of California at Santa Cruz.